# Slice Normalized Dynamic Markov Logic Networks

**Tivadar Papai    Henry Kautz    Daniel Stefankovic**
Department of Computer Science
University of Rochester
Rochester, NY 14627
{papai,kautz,stefanko}@cs.rochester.edu

## Abstract

Markov logic is a widely used tool in statistical relational learning, which uses a weighted first-order logic knowledge base to specify a Markov random field (MRF) or a conditional random field (CRF). In many applications, a Markov logic network (MLN) is trained in one domain, but used in a different one. This paper focuses on dynamic Markov logic networks, where the size of the discretized time-domain typically varies between training and testing. It has been previously pointed out that the marginal probabilities of truth assignments to ground atoms can change if one extends or reduces the domains of predicates in an MLN. We show that in addition to this problem, the standard way of unrolling a Markov logic theory into a MRF may result in time-inhomogeneity of the underlying Markov chain. Furthermore, even if these representational problems are not significant for a given domain, we show that the more practical problem of generating samples in a sequential conditional random field for the next slice relying on the samples from the previous slice has high computational cost in the general case, due to the need to estimate a normalization factor for each sample. We propose a new discriminative model, *slice normalized dynamic Markov logic networks (SN-DMLN)*, that suffers from none of these issues. It supports efficient online inference, and can directly model influences between variables within a time slice that do not have a causal direction, in contrast with fully directed models (*e.g.*, DBNs). Experimental results show an improvement in accuracy over previous approaches to online inference in dynamic Markov logic networks.

## 1   Introduction

Markov logic [1] is a language for statistical relational learning, which employs weighted first-order logic formulas to compactly represent a Markov random field (MRF) or a conditional random field (CRF). A Markov logic theory where each predicate can take an argument representing a time point is called a dynamic Markov logic network (DMLN). We will focus on two-slice dynamic Markov logic networks, *i.e.*, ones in which each quantified temporal argument is of the form $t$ or $t + 1$, in the conditional (CRF) setting. DMLNs are the undirected analogue of dynamic Bayesian networks (DBN) [13] and akin to dynamic conditional random fields [19].

DMLNs have been shown useful for relational inference in complex dynamic domains; for example, [17] employed DMLNs for reasoning about the movements and strategies of 14-player games of Capture the Flag. The usual method for performing offline inference in a DMLN is to simply unroll it into a CRF and employ a general MLN or CRF inference algorithm. We will show, however, that the standard unrolling approach has a number of undesirable properties.

The first two negative properties derive from the fact that MLNs are in general sensitive to the number of constants in each variable domain [6]; and so, in particular cases, unintuitive results can occur when the length of training and testing sequences differ. First, as one increases the number of time points in the domain, the marginals can fluctuate, even if the observations have little or no influence on the hidden variables. Second, the model can become time-inhomogeneous, even if the ground weighted formulas between the time slices originate from the same weighted first-order logic formulas.

The third negative property is of greater practical concern. In domains where there are a large number of variables within each slice dynamic programming based exact inference cannot be used. When

the number of time steps is high and/or online inference is required, unrolling the entire sequence (perhaps repeatedly) becomes prohibitively expensive. Kersting *et al.* [7] suggests reducing the cost by exploiting symmetries while Nath & Domingos [14] propose reusing previously sent messages while performing a loopy belief propagation. Both algorithms are restricted by the capabilities of loopy belief propagation, which can fail to converge to the correct distribution in MLNs. Geier & Biundo [2] provides a slice-by-slice approximate inference algorithm for DMLNs that can utilize any inference algorithm as a black box, but assumes that projecting the distribution over the random variables at every time step to the product of their marginal distributions does not introduce a large degree of error — an assumption that does not always hold. Sequential Monte Carlo methods, or particle filters, are perhaps the most popular methods for online inference in high-dimensional sequential models. However, except for special cases such as, *e.g.*, the Gaussian distributions used in [11], sampling from a two-slice CRF model can become expensive, due to the need to evaluate a partition function for each particle (see Sec. 3 for more details).

As a solution to all of these concerns, we propose a novel way of unrolling a Markov logic theory such that in the resulting probabilistic model a smaller CRF is embedded into a larger CRF making the clique potentials between adjacent slices normalized. We call this model slice normalized dynamic Markov logic network (SN-DMLN). Because of the embedded CRF and the undirected components in our proposed model, the distribution represented by a SN-DMLN cannot be compactly captured by conventional chain graph [10], DBN or CRF graph representations, as we will explain in Sec. 4. The SN-DMLN has none of the negative theoretical or practical properties outlined above, and for accuracy and/or speed of inference matches or outperforms unrolled CRFs and the slice-by-slice approximate inference methods. Finally, because the maximum likelihood parameter learning for an SN-DMLN can be a non-convex optimization problem, we provide an effective heuristic for weight learning, along with initial experimental results.

## 2  Background

Probabilistic graphical models compactly represent probability distributions using a graph structure that expresses conditional independences among the variables. Directed graphical models are mainly used in the generative setting, *i.e.*, they model the joint distribution of the hidden variables and the observations, and during training the joint probability of the training data is maximized. Hidden Markov models are the prototypical directed models used for sequential data with hidden and observable parts. It has been demonstrated that for classification problems, discriminative models, which model the conditional probability of the hidden variables given the observations, can outperform generative models [12]. The main justifications for their success are that complex dependencies between observed variables do not have to be modeled explicitly, and the conditional probability of the training data (which is maximized during parameter learning) is a better objective function if we eventually want to use our model for classification. Markov random fields (MRFs) and conditional random fields (CRFs) belong to the class of undirected graphical models. MRFs are generative models, while CRFs are their discriminative version. (For a more detailed discussion of the relationships between these models see [8]). *Markov logic* [1] is a first-order probabilistic language that allows one to define template features that apply to whole classes of objects at once. A *Markov logic network* is a set of weighted first-order logic formulas and a finite set of constants $C = \{c_1, c_2, \ldots, c_{|C|}\}$ which together define a Markov network $M_{L,C}$ that contains a binary node for each possible grounding of each predicate (ground atom) and a binary valued feature for each grounding of each first-order logic formula. We will also call the ground atoms variables (since they are random variables). In each truth assignment to the variables, each variable or feature (ground formula) evaluates to 1 (true) or 0 (false). In this paper we assume function-free clauses and Herbrand interpretations. Using the knowledge base we can either create an MRF or a CRF. If we instantiate the model as a CRF, the conditional probability of a truth assignment $y$ to the hidden ground atoms (*query atoms*) in an MLN, given truth assignment $x$ to the observable ground atoms (*evidence atoms*), is defined as:

$$Pr(Y = y | X = x) = \frac{\exp(\sum_i w_i \sum_j f_{i,j}(x, y))}{Z(x)}, \qquad (1)$$

where $f_{i,j}(x, y) = 1$ if the $j$th grounding of the $i$th formula is true under truth assignment $\{x, y\}$, and $f_{i,j}(x, y) = 0$ otherwise. $w_i$ is the weight of the $i$th formula and $Z(x)$ is the normalization factor. Ground atoms share the same weight if they are groundings of the same weighted first-order logic formula, and (1) could be expressed in terms of $n_i(x, y) = \sum_j f_{i,j}(x, y)$. Instantiation as an MRF can be done similarly, having an empty set of evidence atoms. Dynamic MLNs [7] are MLNs with distinguished arguments in every predicate representing the flow of time or some other sequential quantity. In our settings, $Y_t$ and $X_t$ will denote the set of hidden and observable random variables, respectively, at time $t$, and $Y_{1:t}$ and $X_{1:t}$ from time step 1 to $t$. Each set can contain many variables, and we should note that their distribution will be represented compactly by weighted first-order logic formulas. The formulas in the knowledge base can be partitioned into

two sets. The *transitions* part contains the formulas for which it is true that for any grounding of each formula, there is a $t$ such that the grounding shares variables only with $Y_t$ and $Y_{t+1}$. The *emission* part represents the formulas which connect the hidden and observable variables, i.e. $Y_t$ and $X_t$. We will use $\tilde{P}(Y_t, Y_{t+1})$ (or $\tilde{P}(Y_{t:t+1})$) and $\tilde{P}(Y_t, X_t)$ to denote the product of the potentials corresponding to weighted ground formulas at time $t$ of the transition and the observation formulas, respectively. Since some ground formulas may contain only variables from $Y_t$ ( *i.e.*, defined over hidden variables within the same slice), in order to count the corresponding potentials exactly once, we always include their potentials $\tilde{P}(Y_t, Y_{t-1})$, and for $t = 1$ we have a separate $\tilde{P}(Y_1)$. Hence, the distribution defined in (1) in sequential domains can be factorized as:

$$Pr(Y_{1:t} = y_{1:t} | X_{1:t} = x_{1:t}) = \frac{\tilde{P}_1(Y_1 = y_1) \prod_{i=2}^{t} \tilde{P}(Y_{i-1:i} = y_{i-1:i}) \prod_{i=1}^{t} \tilde{P}(Y_i = y_i, X_i = x_i)}{Z(x_{1:t})}$$

(2)

In the rest of the paper, we only allow the temporal domain to vary, and the rest of the domains are fixed.

## 3    Unrolling MLNs into random fields in temporal domains

We now describe disadvantages of the standard definition of DMLNs, *i.e.*, when the knowledge base is unrolled into a CRF:

1. As one increases the number of time points the marginals can fluctuate, even if all the clique potentials $\tilde{P}(Y_i = y_i, X_i = x_i)$ in (2) are uninformative.

2. The transition probability $\Pr(Y_{i+1}|Y_i)$ can be dependent on $i$, even if every $\tilde{P}(Y_i = y_i, X_i = x_i)$ is uninformative and we use the same weighted first-order logic formula responsible for the ground formulas covering the transitions between every $i$ and $i + 1$.

3. Particle filtering is *costly* in general, *i.e.*, if we have the marginal probabilities at time $t$, we cannot compute them at time $t + 1$ using particle filtering unless certain special conditions are satisfied.

Saying that $\tilde{P}(Y_i = y_i, X_i = x_i)$ is uninformative is equivalent to saying that $\tilde{P}(Y_i = y_i, X_i = x_i)$ is constant. (Note that, if $Y_i$ and $X_i$ are independent, *i.e.*, for some $q$ and $r$ $\tilde{P}(Y_i = y_i, X_i = x_i) = r(y_i)q(x_i)$ then $q$ could be marginalized out and $r(Y_i)$ could be snapped to $\tilde{P}(Y_i, Y_{i-1})$ in (2).) To demonstrate Property 1, consider an unrolled MRF with the temporal domain $\mathcal{T} = \{1, \ldots, T\}$, with only predicate $P(t)$ ($t \in \mathcal{T}$) and with the weighted formulas $(+\infty, P(t) \Leftrightarrow P(t + 1))$ (hard constraint) and $(w, P(t))$ (soft constraint). Because of the hard constraint, only the sequences $\forall t : P(t)$ and $\forall t : \neg P(t)$ have non-zero probabilities. The soft weights imply that $\Pr(P(t)) = \exp(wT)\Pr(\neg P(t))$, *i.e.*, $\Pr(P(t))$ converges to $1, 0$ or to $0.5$ with exponential rate depending on the sign of $w$. But we are not always fortunate to have converging marginals, *e.g.*, if we change the hard constraint to be $P(t) \Leftrightarrow \neg P(t + 1)$ and $w \neq 0$ the marginals will diverge. If $T$ is even, then for every $t \in \mathcal{T}$, $\Pr(P(t)) = \Pr(\neg P(t))$, since in both sequences $P(t)$ has the same number of true groundings. If $T$ is odd then for every odd $t \in \mathcal{T}$: $\Pr(P(t)) = \exp(w)\Pr(\neg P(t))$. Consequently, we have diverging marginals as $T \to +\infty$. This phenomenon not only makes the inference unreliable, but a weight learning algorithm that maximizes the log-likelihood of the data would produce different weights depending on whether $T$ is even or odd. A similar effect arising from moving between different sized domains is discussed in more details in [6]. The akin Property 2 (inhomogeneity) can be demonstrated similarly, consider, *e.g.*, an MLN with a single first-order logic formula $P(t) \vee P(t + 1)$ with weight $w$. For the sake of simplicity, assume $T = 3$. The unrolled MRF defines a distribution where $\Pr(\neg P(3)|\neg P(2)) = \frac{1 + exp(w)}{1 + 2exp(w) + \exp(2w)}$ which is not equal to $\Pr(\neg P(2)|\neg P(1)) = \frac{1 + exp(w)}{1 + exp(w) + 2\exp(2w)}$ for an arbitrary choice of $w$.

The examples we just gave involved hard constraints. In fact, we can show that if there are no hard hard constraints, as $T$ increases the marginals converge and the system becomes homogeneous (except for a finite number of transitions). Consider the matrix $\Phi$ *s.t.* $\Phi_{i,j} = \tilde{P}(Y_t = a_j, Y_{t-1} = a_i)$, where $a_i, i = 1, \ldots, N$ is an enumeration of the all the possible truth assignments within each slice and $N$ is the number of the possible truth assignments in the slice. Let $\Pr_T(Y_1 = y_1) = \frac{1}{Z(Y_{1:T})} \sum_{y_2, \ldots, y_T} \prod_{i=1}^{T-1} \tilde{P}(Y_i = y_i, Y_{i+1} = y_{i+1})$, where $Z(Y_{1:T}) = \sum_{y_1, \ldots, y_T} \prod_{i=1}^{T-1} \tilde{P}(Y_i = y_i, Y_{i+1} = y_{i+1})$.

**Proposition 1.** $\lim_{t \to \infty} Pr_t(Y_1 = y)$ *exists if $\Phi$ is a positive matrix, i.e., $\forall i, j : \Phi_{i,j} > 0$.*

*Proof.* Using $\Phi$ and the notation $\vec{1}$ for all one vector and $\vec{e_i}$ for a vector which has $1$ at the $i$th component and $0$ everywhere else, we can express $\Pr_t(Y_1 = y)$ as:

$$\Pr_t(Y_1 = y) = \frac{\sum_{y_2} \tilde{P}(Y_1 = a_i, Y_2 = y_2)\vec{e_i}\Phi^{t-1}\vec{1}}{\vec{1}^T \Phi^t \vec{1}} \tag{3}$$

Since $\Phi$ is positive we can apply theorem 8.2.8. from [5], *i.e.*, if the spectral radius of $\Phi$ is $\rho(\Phi)$ (which is always positive for positive matrices): $\lim_{t\to\infty}(\rho^{-1}(\Phi)\Phi)^t = L$, where $L = xy^T$, $\Phi x = \rho(\Phi)x$, $\Phi^T y = \rho(\Phi)y$, $x > 0, y > 0$ and $x^T y = 1$. Dividing both the numerator and the denominator by $\rho^t(\Phi)$ in (3) proves the convergence of $\Pr_t(Y_1 = y)$. $\qquad\square$

The issue of diverging marginals and time-inhomogeneity has not been previously recognized as a practical problem. However, the increasing interest in probabilistic models that contain large numbers of deterministic constraints (see, *e.g.* [4]) might bring this issues to the fore. This proposition can serve as an explanation why in practice we do not encounter diverging marginals in linear chain type CRFsand why except for a finite number of transitions the model becomes time-homogeneous.

A more significant practical challenge is described by Property 3, the problem of sampling from $\Pr(Y_t|X_{1:t} = x_{1:t})$ using the previously drawn samples from $\Pr(Y_{t-1}|X_{1:t-1} = x_{1:t-1})$. In a directed graphical model (*e.g.*, in a hidden Markov model), following standard particle filter design, having sampled $s_{1:t-1} \sim \Pr(Y_{1:t-1} = s_{1:t-1}|X_{1:t-1} = x_{1:t-1})$, and then using $s_{1:t-1}$ one would sample $s_t \sim \Pr(Y_t, Y_{1:t-1} = s_{1:t-1}|X_{1:t-1})$. Since

$$\Pr(Y_{1:t} = s_{1:t}|X_{1:t-1} = x_{1:t-1}) = \Pr(Y_t = s_t|Y_{t-1} = s_{t-1})\Pr(Y_{1:t-1} = s_{1:t-1}|X_{1:t-1} = x_{1:t-1}) \tag{4}$$

we do not have any difficulty performing this sampling step, and all that is left is to re-sample the collection of $s_{1:t}$ with importance weights $\Pr(Y_t = s_t|X_t = x_t)$. The analogue of this process does not work in a CRF in general. If one first draws a sample $s_{1:t-1} \sim \tilde{P}(Y_1, X_1 = x_1)\tilde{P}(Y_1)\prod_{i=2}^{t-1} \tilde{P}(Y_i, Y_{i-1})\tilde{P}(Y_i, X_i = x_i)$, and then draws $s_t \sim \tilde{P}(Y_t, Y_{t-1} = s_{t-1})$, we end up sampling from:

$$s \sim \tilde{P}(Y_1, X_1 = x_1)\tilde{P}(Y_1)\prod_{i=2}^{t} \tilde{P}(Y_i, Y_{i-1})\tilde{P}(Y_i, X_i = x_i)\frac{1}{Z_{t-1}(y_{t-1})} \tag{5}$$

where $Z_{t-1}(y_{t-1}) = \sum_{y_t} \tilde{P}(Y_t = y_t, Y_{t-1} = y_{t-1})$. Unless $Z_{t-1}(y_{t-1})$ is the same for every $y_{t-1}$, it is necessary to approximate $Z_{t-1}(s_{t-1})$ for every $s_{t-1}$. [1] Although several algorithms have been proposed to estimate partition functions [16, 18], the partition function estimation can increase both the running time of the sampling algorithm significantly and the error of the approximation of the sampling algorithm. While there are restricted special cases where the normalization factor can be ignored [11], in general ignoring the approximation of $Z_{t-1}(y_{t-1})$ could cause a large error in the computed marginals. Consider, *e.g.*, when we have three weighted formulas in the previously used toy domain, namely, $w : \neg P(Y_t) \vee \neg P(Y_{t+1})$, $-w : P(Y_t) \wedge \neg P(Y_{t+1})$ and $w' : P(Y_t) \leftrightarrow \neg P(Y_{t+1})$, where $w > 0$ and $w' < 0$. It can be proved that in this setting using particle filtering in a CRF without accounting for $Z_{t-1}(y_{t-1})$ would result in $\lim_{t\to\infty} \Pr(P(Y_t)) = \frac{1}{2}$, while in the CRF the correct marginal would be $\lim_{t\to\infty} \Pr(P(Y_t)) = 1 - \frac{\exp(w)}{1+\exp(w)}\exp(w') + \mathcal{O}(\exp(2w'))$, which gets arbitrarily close to 1 as we decrease $w'$.

## 4 Slice normalized DMLNs

As we demonstrated in Section 3, the root cause of the weaknesses of an ordinarily unrolled CRF lies in that $\tilde{P}(Y_t = y_t, Y_{t-1} = y_{t-1})$ is unnormalized, *i.e.*, $\sum_{y_t} \tilde{P}(Y_t = y_t, Y_{t-1} = y_{t-1}) \neq 1$ in general. One approach to introduce normalization could be to use maximum entropy Markov models (MEMM) [12]. In that case we would directly represent $\Pr(Y_t|X_t, Y_{t-1})$, hence we could implement a sequential Monte Carlo algorithm simply directly sampling $s_t \sim \Pr(Y_t|X_t = x_t, Y_{t-1} = s_{t-1})$ from slice to slice. However, in [9], it was pointed out that MEMMs suffer from the *label-bias* problem to which as a solution CRFs were invented. Chain graphs (see *e.g.* [10]) have also the advantage of mixing directed and undirected components, and would be a tempting choice to use, but they could only model the transition between slices by either representing (i) $\Pr(Y_t|X_t = x_t, Y_{t-1} = s_{t-1})$,

in which case the model would again suffer from the label-bias problem, or (ii) $\Pr(Y_t, X_t|Y_{t-1})$ or (iii) $\Pr(X_t|Y_t)$ and $\Pr(Y_t|Y_{t-1})$. The defined distributions both in (ii) and (iii) do not give any advantage performing the sampling step in (4), and similarly to CRFs would require the expensive computation of a normalization factor. We propose a slice normalized dynamic Markov logic network (SN-DMLN) model, which consists of directed and undirected components on the high level, and can be thought of as a smaller CRF nested into a larger CRF describing the transition probabilities constructed using weighted first-order logic formulas as templates. SN-DMLNs neither suffer from the label bias problem, nor bear the disadvantageous properties presented in Section 3. The distribution defined by an unrolled SN-DMLN is as follows:

$$\Pr(Y_{1:t} = y_{1:t}|X_{1:t} = x_{1:t}) = \frac{1}{Z(x_{1:t})} P_1(Y_1) \prod_{i=1}^{t} \tilde{P}(Y_i = y_i, X_i = x_i) \tag{6}$$

$$\prod_{i=2}^{t} P(Y_i = y_i|Y_{i-1} = y_{i-1}) \,,$$

where

$$P_1(Y_1 = y_1) = \frac{\tilde{P}(Y_1 = y_1)}{\sum_{y_1'} \tilde{P}(Y_1 = y_1')} \,, \quad P(Y_i = y_i|Y_{i-1} = y_{i-1}) = \frac{\tilde{P}(Y_i = y_i, Y_{i-1} = y_{i-1})}{\sum_{y_i'} \tilde{P}(Y_i = y_i', Y_{i-1} = y_{i-1})} \,,$$

and the partition function is defined by:

$$Z(x_{1:t}) = \sum_{y_1,\ldots,y_t} \left\{ P_1(Y_1 = y_1) \prod_{i=1}^{t} \tilde{P}(Y_i = y_i, X_i = x_i) \prod_{i=2}^{t} P(Y_i = y_i|Y_{i-1} = y_{i-1}) \right\} .$$

$P(Y_t = y_t|Y_{t-1} = y_{t-1})$ is defined by a two-slice Markov logic network (CRF), which describes the state transitions probabilities in a compact way. If we hide the details of this nested CRF component and treat it as one potential, we could represent the distribution in (6) by regular chain graphs or CRFs; however we would lose then the compactness the nested CRF provides for describing the distribution. Similarly, we could collapse the variables at every time slice into one and could use a DBN (or again a chain graph), but it would need exponentially more entries in its conditional probability tables. If $\tilde{P}(Y_i = y_i, X_i = x_i)$ does not have any information content , the probability distribution defined in (6) reduces to $P_1(Y_1 = y_1) \prod_{i=2}^{t} P(Y_i = y_i|Y_{i-1} = y_{i-1})$, which is a time-homogeneous Markov chain [2] , hence this model clearly does not have Properties 1 and 2, no matter what formulas are present in the knowledge base. Furthermore, we do not have to compute the partition function between the slices, because equation (5) shows, drawing a sample $y_t \sim \tilde{P}(Y_t, Y_{t-1} = y_{t-1})$ while keeping the value $y_{t-1}$ fixed is equivalent to sampling from $P(Y_t|Y_{t-1} = y_{t-1})$, the quantity present in equation (6). This means that using our model one can avoid estimating $Z(y_{t-1})$. To learn the parameters of the model we will maximize the conditional log-likelihood ($\mathcal{L}$) of the data. We use a modified version of a hill climbing algorithm. The modification is needed, because in our case $\mathcal{L}$ is not necessarily concave. We will partition the weights (parameters) of our model based on whether they belong to transition or to emission part of the model. The gradient of the $\mathcal{L}$ of a data sequence $d = y_1, x_1, \ldots, y_t, x_t$ w.r.t. an emission parameter $w_e$ (to which feature $n_e$ belongs) is:

$$\frac{\partial \mathcal{L}_d}{\partial w_e} = \sum_{i=1}^{t} n_e(y_i, x_i) - \mathbb{E}_{Pr(Y|X=x)} \left[ \sum_{i=1}^{t} n_e(Y_i, x_i) \right] \,, \tag{7}$$

which is analogous to what one would expect for CRFs. However, for a transition parameter $w_{tr}$ (belonging to feature $n_{tr}$) we get something different:

$$\frac{\partial \mathcal{L}_d}{\partial w_{tr}} = \sum_{i=1}^{t} n_{tr}(y_{i+1}, y_i) - \sum_{i=1}^{t-1} \mathbb{E}_{P(Y_{i+1}|y_i)} [n_{tr}(Y_{i+1}, Y_i = y_i)] \tag{8}$$

$$- \mathbb{E}_{Pr(Y|X=x)} \left[ \sum_{i=1}^{t-1} n_{tr}(Y_{i+1}, Y_i) - \sum_{i=1}^{t-1} \mathbb{E}_{P(\tilde{Y}_{i+1}|Y_i)} \left[ n_{tr}(\tilde{Y}_{i+1}, Y_i) \right] \right] .$$

(Note that, $\mathcal{L}_d$ is concave *w.r.t.* the emission parameters, *i.e.*, when the transition parameters are kept fixed, allowing the transition parameters to vary makes $\mathcal{L}_d$ no longer concave.) In (8) the first

| friendships reflect people's similarity in smoking habits | $Smokes(p_1,t) \land \neg Smokes(p_2,t) \land (p_1 \neq p_2) \supset \neg Friends(p_1,p_2,t)$ |
|---|---|
| | $Smokes(p_1,t) \land Smokes(p_2,t) \land (p_1 \neq p_2) \supset Friends(p_1,p_2,t)$ |
| | $\neg Smokes(p_1,t) \land \neg Smokes(p_2,t) \land (p_1 \neq p_2) \supset Friends(p_1,p_2,t)$ |
| symmetry and reflexivity of friendship | $\neg Friends(p_1,p_2,t) \supset \neg Friends(p_2,p_1,t)$ |
| | $Friends(p_1,p_2,t) \supset Friends(p_2,p_1,t)$ |
| | $Friends(p,p,t)$ |
| persistence of smoking | $Smokes(p,t) \supset Smokes(p,t+1)$ |
| | $\neg Smokes(p,t) \supset \neg Smokes(p,t+1)$ |
| people with different smoking habits hang out separately | $Hangout(p_1,g_1,t) \land Hangout(p_2,g_2,x) \land Smokes(p_1,t) \land$ $(p_1 \neq p_2) \land (g_1 \neq g_2) \supset \neg Smokes(p_2,t)$ |
| | $Hangout(p_1,g_1,t) \land Hangout(p_2,g_2,t) \land \neg Smokes(p_1,t) \land$ $(p_1 \neq p_2) \land (g_1 \neq g_2) \supset Smokes(p_2,t)$ |

Table 1: Formulas in the knowledge base

two and the last two terms can be grouped together. The first group would represent the gradient in the case of uninformative observations, *i.e.*, when the model simplifies to a Markov chain with a compactly represented transition probability distribution. The second group is the expected value of the expression in the first group. The first three terms correspond to the gradient of a concave function; while the fourth term corresponds to the gradient of a convex function, so the function as a whole is not guaranteed to be maximized by convex optimization techniques alone. Therefore, we chose a heuristic for our optimization algorithm which gradually increases the effects of the second group in the gradient. More precisely, we always compute the gradient w.r.t. $w_o$ according to (7), but w.r.t. $w_{tr}$ we use:

$$\frac{\partial \mathcal{L}_d}{\partial w_{tr}} = \sum_{i=1}^{t} n_{tr}(y_{i+1},y_i) - \sum_{i=1}^{t-1} \mathbb{E}_{P(Y_{i+1}|y_i)} \left[ n_{tr}(Y_{i+1},y_i) \right] \tag{9}$$

$$- \alpha \mathbb{E}_{Pr(Y|X=x)} \left[ \sum_{i=1}^{t} n_{tr}(Y_{i+1},Y_i) - \sum_{i=1}^{t-1} \mathbb{E}_{P(\tilde{Y}_{i+1}|Y_i)} \left[ n_{tr}(\tilde{Y}_{i+1},Y_i) \right] \right]$$

where $\alpha$ is kept at the value of $0$ until convergence, and then gradually increased from $0$ to $1$ to converge to the nearest local optimum. In Section 5, we experimentally demonstrate that this heuristic provides reasonably good results, hence we did not turn to more sophisticated algorithms. The rationale behind our heuristic is that if $\tilde{P}(Y_i = y_i, X_i = x_i)$ had truly no information content, then for $\alpha = 0$ we would find the global optimum, and as we increase $\alpha$ we are taking into account that the observations are correlated with the hidden variables with an increasing weight.

# 5 Experiments

For our experiments we extended the Probabilistic Consistency Engine (PCE) [3], a Markov logic implementation that has been used effectively in different problem domains. For training, we used 10000 samples for the unrolled CRF and 100 particles and 100 samples for approximating the conditional expectations in (9) for the SN-DMLN to estimate the gradients. For inference we used 10000 samples for the CRF and 10000 particles for the mixed model. The sampling algorithm we relied on was MC-SAT [15]. Our example training data set was a modified version of the dynamic social network example [7, 2]. The hidden predicates in our knowledge base were $Smokes(person,time), Friends(person_1,person_2,time)$ and the observable was $Hangout(person,group,time)$. The goal of inference was to predict which people could potentially be friends, based on the similarity in their smoking habits, which similarity could be inferred based on the groups the individuals hang out. We generated training and test data as follows: there were two groups $g_1$, $g_2$, one for smokers and one for non-smokers. Initially 2 people were randomly chosen to be smokers and 2 to be non-smokers. People with the same smoking habits can become friends at any time step with probability $1 - 0.05\alpha$, and a smoker and a non-smoker can become friends with probability $0.05\alpha$. Every 5th time step (starting with $t = 0$) people hang out in groups and for each person the probability of joining one of the groups is $1 - 0.05\alpha$. With probability $1 - 0.05\alpha$, everyone spends time with the group reflecting their smoking habits, and with probability $0.05\alpha$ they go to hang out with the other group. The rest of the days people do not hang out. The smoking habits persist, *i.e.*, a smoker stays a smoker and a non-smoker stays a non-smoker at the next time step with probability $1 - 0.05\alpha$. In our two configurations we had $\alpha = 0$ (deterministic case) and $\alpha = 1$ (non-deterministic case). The weights of the clauses we learned using the SN-DMLN and the CRF unrolled models are in Table 1.

We used chains with length 5, 10, 20 and 40 as training data, respectively. For each chain we had 40, 20, 10 and 5 examples both for the training and for testing, respectively. In our experiments we compared three types of inference, and measured the prediction quality for the hidden predicate $Friends$ by assigning true to every ground atom the marginal probability of which was greater than

| length | $\alpha = 0$ | | | | | | $\alpha = 1$ | | | | | |
|---|---|---|---|---|---|---|---|---|---|---|---|---|
| | accuracy | | | f1 | | | accuracy | | | f1 | | |
| | SN | MAR | MC-SAT | SN | MAR | MC-SAT | SN | MAR | MC-SAT | SN | MAR | MC-SAT |
| 5 | 1.0 | 0.40 | 1.0 | 1.0 | 0.14 | 1.0 | 0.84 | 0.36 | 0.81 | 0.75 | 0.10 | 0.69 |
| 10 | 1.0 | 0.40 | 0.97 | 1.0 | 0.14 | 0.95 | 0.84 | 0.36 | 0.77 | 0.74 | 0.11 | 0.61 |
| 20 | 1.0 | 0.40 | 0.67 | 1.0 | 0.14 | 0.49 | 0.92 | 0.55 | 0.66 | 0.85 | 0.32 | 0.47 |
| 40 | 1.0 | 0.85 | 0.60 | 1.0 | 0.72 | 0.43 | 0.88 | 0.73 | 0.59 | 0.78 | 0.55 | 0.42 |

Table 2: Accuracy and F-score results when models were trained and tested on chains with the same length

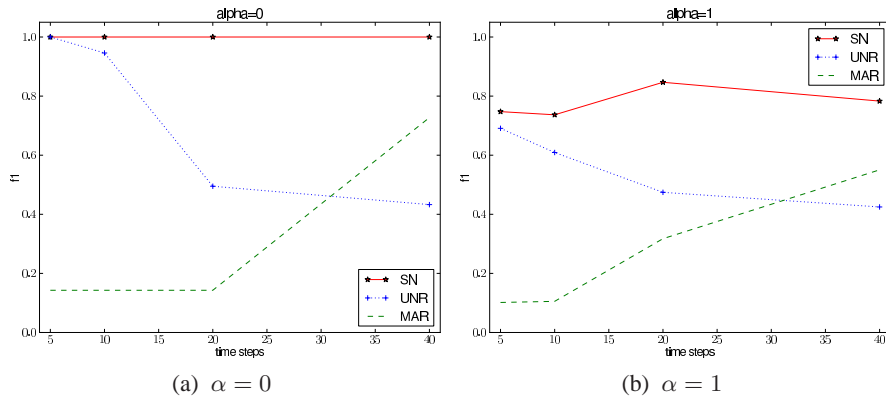

(a) $\alpha = 0$             (b) $\alpha = 1$

Figure 1: F-score of models trained and tested on the same length of data

$0.55$, and false if its probability was less than $0.45$; otherwise we considered it as a misclassification. Prediction of $Smokes$ was impossible in the generated data set, because the data generation was symmetric *w.r.t* to smoking and not smoking, and from the observations we could only tell that certain pairs of people have similar or different smoking habits, but not who smokes and who does not. The three methods we compared were (i) particle filtering in the SN-DMLN model *(SN)*, (ii) the approximate online inference algorithm of [2], which projects the inferred distribution of the random variables at the previous slice to the product of their marginals, and incorporates this information into a two slice MLN to infer the probabilities at the next slice (we re-implemented the algorithm in PCE) *(MAR)*, and (iii) using a general inference algorithm (MC-SAT [15]) for a CRF which is always completely unrolled in every time step *(UNR)*. In UNR and MAR the same CRF models were used. The training of the SN-DMLN model took approximately for 120 minutes for all the test cases, while for the CRF model, it took 120, 145, 175 and 240 minutes respectively. The inference over the entire test set, took approximately 6 minutes for SN and MAR in every test case, while UNR required 5, 8, 12 and 40 minutes for the different test cases. The accuracy and F-scores for the different test cases are summarized in Table 2 and the F-scores are plotted in Fig. 1.

SN outperforms MAR, because as we see that in the knowledge base, MAR can only conclude that people have the same or different smoking habits on the days when people hang out (every 5th time step), and the marginal distributions of $Smokes$ do not carry enough information about which pair of people have different smoking habits, hence the quality of MAR's prediction decreases on days when people do not hang out. The performance of SN and MAR stays the same as we increase the length of the chain while the performance of UNR degrades. This is most pronounced in the deterministic case ($\alpha = 0$). This can be explained by that MC-SAT requires more sampling steps to maintain the same performance as the chain length increases.

To demonstrate that if we use the same number of particles in SN as number of samples in UNR, the performance of SN stays approximately the same while the performance of UNR degrades over time, we trained both the CRF and SN-DMLN on length 5 chains where both SN and UNR were performing equally well and used test sets of different lengths up to length 150. F-scores are plotted in Fig. 2.

We see from Fig. 2 that SN outperforms both UNR and MAR as the chain length increases. Moreover, UNR's performance is clearly decreasing as the length of the chain increases.

## 6  Conclusion

In this paper, we explored the theoretical and practical questions of unrolling a sequential Markov logic knowledge base into different probabilistic models. The theoretical issues arising in a CRF-

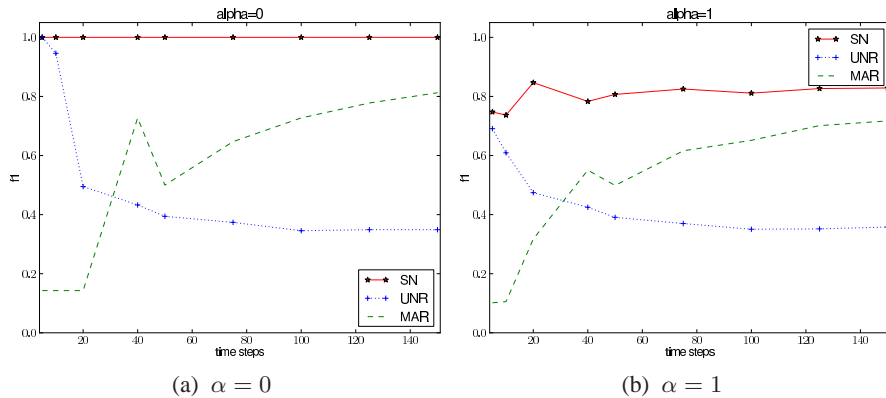

(a) $\alpha = 0$          (b) $\alpha = 1$

Figure 2: F-score of models trained and tested on different length of data

based MLN unrolling are a warning that unexpected results may occur if the observations are weakly correlated with the hidden variables. We gave a qualitative justification why this phenomenon is more of a theoretical concern in domains lacking deterministic constraints. We demonstrated that the CRF based unrolling can be outperformed by a model that mixes directed and undirected components (the proposed model does not suffer from any of the theoretical weaknesses, nor from the label-bias problem).

From a more practical point of view, we showed that our proposed model provides computational savings, when the data has to be processed in a sequential manner. These savings are due to that we do not have to unroll a new CRF at every time step, or estimate a partition function which is responsible for normalizing the product of clique potentials appearing in two consecutive slices. The previously used approximate inference methods in dynamic MLNs either relied on belief propagation or assumed that approximating the distribution at every time step by the product of the marginals would not cause any error. It is important to note that, although in our paper we focused on marginal inference, finding the most likely state sequence could be done using the generated particles. Although the conditional log-likelihood of the training data in our model may be non-concave so that hill climbing based approaches could fail to settle in a global maximum, we proposed a heuristic for weight learning and demonstrated that it could train our model so that it performs as well as conditional random fields. Although training the mixed model might have a higher computational cost than training a conditional random field, but this cost is amortized over time, since in applications inference is performed many times, while weight learning only once. Designing more scalable weight learning algorithms is among our future goals.

## 7 Acknowledgments

We thank Daniel Gildea for his insightful comments.

This research was supported by grants from ARO (W991NF-08-1-0242), ONR (N00014-11-10417), NSF (IIS-1012017), DOD (N00014-12-C-0263), and a gift from Intel.

## Footnotes

[1] Exploiting inner structure according to the graphical model within the slice would in worst case still result in computation of the expensive partition function, or could result in a higher variance estimator the same way as, *e.g.*, using a uniform proposal distribution does.

[2]Note that, in the SN-DMLN model the uniformity of $\tilde{P}(Y_i = y_i, X_i = x_i)$ is a stronger assumption than the independence of $X_i$ and $Y_i$.

## References

[1] Pedro Domingos and Daniel Lowd. *Markov Logic: An Interface Layer for Artificial Intelligence*. Synthesis Lectures on Artificial Intelligence and Machine Learning. Morgan & Claypool Publishers, 2009.

[2] Thomas Geier and Susanne Biundo. Approximate online inference for dynamic markov logic networks. In *Tools with Artificial Intelligence (ICTAI), 2011 23rd IEEE International Conference on*, pages 764–768, 2011.

[3] Shalini Ghosh, Natarajan Shankar, and Sam Owre. Machine reading using markov logic networks for collective probabilistic inference. In *In Proceedings of ECML-CoLISD.*, 2011.

[4] Vibhav Gogate and Rina Dechter. Samplesearch: Importance sampling in presence of determinism. *Artif. Intell.*, 175(2):694–729, 2011.

[5] Roger A. Horn and Charles R. Johnson. *Matrix Analysis*. Cambridge University Press, 1990.

[6] Dominik Jain, Andreas Barthels, and Michael Beetz. Adaptive Markov Logic Networks: Learning Statistical Relational Models with Dynamic Parameters. In *19th European Conference on Artificial Intelligence (ECAI)*, pages 937–942, 2010.

[7] K. Kersting, B. Ahmadi, and S. Natarajan. Counting belief propagation. In J. Bilmes A. Ng, editor, *Proceedings of the 25th Conference on Uncertainty in Artificial Intelligence (UAI–09)*, Montreal, Canada, June 18–21 2009.

[8] D. Koller and N. Friedman. *Probabilistic Graphical Models: Principles and Techniques*. MIT Press, 2009.

[9] John Lafferty. Conditional random fields: Probabilistic models for segmenting and labeling sequence data. pages 282–289. Morgan Kaufmann, 2001.

[10] Steffen Lauritzen and Thomas S. Richardson. Chain graph models and their causal interpretations. *B*, 64:321–361, 2001.

[11] B. Limketkai, D. Fox, and Lin Liao. CRF-Filters: Discriminative Particle Filters for Sequential State Estimation. In *Robotics and Automation, 2007 IEEE International Conference on*, pages 3142–3147, 2007.

[12] Andrew McCallum, Dayne Freitag, and Fernando C. N. Pereira. Maximum entropy markov models for information extraction and segmentation. In *Proceedings of the Seventeenth International Conference on Machine Learning*, ICML '00, pages 591–598, San Francisco, CA, USA, 2000. Morgan Kaufmann Publishers Inc.

[13] Kevin Patrick Murphy. *Dynamic bayesian networks: representation, inference and learning*. PhD thesis, 2002. AAI3082340.

[14] Aniruddh Nath and Pedro Domingos. Efficient belief propagation for utility maximization and repeated inference, 2010.

[15] Hoifung Poon and Pedro Domingos. Sound and efficient inference with probabilistic and deterministic dependencies. In *Proceedings of the 21st national conference on Artificial intelligence - Volume 1*, AAAI'06, pages 458–463. AAAI Press, 2006.

[16] G. Potamianos and J. Goutsias. Stochastic approximation algorithms for partition function estimation of Gibbs random fields. *IEEE Transactions on Information Theory*, 43(6):1948–1965, 1997.

[17] Adam Sadilek and Henry Kautz. Recognizing multi-agent activities from GPS data. In *Twenty-Fourth AAAI Conference on Artificial Intelligence*, 2010.

[18] R. Salakhutdinov. Learning and evaluating Boltzmann machines. Technical Report UTML TR 2008-002, Department of Computer Science, University of Toronto, June 2008.

[19] Charles Sutton, Andrew McCallum, and Khashayar Rohanimanesh. Dynamic conditional random fields: Factorized probabilistic models for labeling and segmenting sequence data. *J. Mach. Learn. Res.*, 8:693–723, May 2007.

